# Filter Selection Model for Generating Visual Motion Signals

**Steven J. Nowlan***
CNL, The Salk Institute
P.O. Box 85800, San Diego, CA
92186-5800

**Terrence J. Sejnowski**
CNL, The Salk Institute
P.O. Box 85800, San Diego, CA
92186-5800

## Abstract

Neurons in area MT of primate visual cortex encode the velocity of moving objects. We present a model of how MT cells aggregate responses from V1 to form such a velocity representation. Two different sets of units, with local receptive fields, receive inputs from motion energy filters. One set of units forms estimates of local motion, while the second set computes the utility of these estimates. Outputs from this second set of units "gate" the outputs from the first set through a gain control mechanism. This active process for selecting only a subset of local motion responses to integrate into more global responses distinguishes our model from previous models of velocity estimation. The model yields accurate velocity estimates in synthetic images containing multiple moving targets of varying size, luminance, and spatial frequency profile and deals well with a number of transparency phenomena.

## 1 INTRODUCTION

Humans, and primates in general, are very good at complex motion processing tasks such as tracking a moving target against a moving background under varying luminance. In order to accomplish such tasks, the visual system must integrate many local motion estimates from cells with limited spatial receptive fields and marked orientation selectivity. These local motion estimates are sensitive not just

to the velocity of a visual target, but also to many other features of the target such as its spatial frequency profile or local edge orientation. As a result, the integration of these motion signals cannot be performed in a fixed manner, but must be a dynamic process dependent on the visual stimulus.

Although cells with motion-sensitive responses are found in primary visual cortex (V1 in primates), mounting physiological evidence suggests that the integration of these responses to produce responses which are tuned primarily to the velocity of a visual target first occurs in primate visual area MT (Albright 1992, Maunsell and Newsome 1987). We propose a computational model for integrating local motion responses to estimate the velocity of objects in the visual scene. These velocity estimates may be used for eye tracking or other visuo-motor skills. Previous computational approaches to this problem (Grzywacz and Yuille 1990, Heeger 1987, Heeger 1992, Horn and Schunk 1981, Nagel 1987) have primarily focused on *how* to combine local motion responses into local velocity estimates at all points in an image (the velocity flow field). We propose that the integration of local motion measurements may be much simpler, if one does not try to integrate across all of the local motion measurements but only a subset. Our model learns to estimate the velocity of visual targets by solving the problems of *what* to integrate and *how* to integrate in parallel. The trained model yields accurate velocity estimates from synthetic images containing multiple moving targets of varying size, luminance, and spatial frequency profile.

## 2   THE MODEL

The model is implemented as a cascade of networks of locally connected units which has two parallel processing pathways (figure 1). All stages of the model are represented as "layers" of units with a roughly retinotopic organization. The figure schematically represents the activity in the model at one instant of time. Conceptually, it is easier to think of the model as computing evidence for particular velocities in an image rather than computing velocity directly. Processing in the model may be divided into 3 stages, to be described in more detail below. In the first stage, the input intensity image is converted into 36 local motion "images" (9 of which are shown in the figure) which represent the outputs of 36 motion energy filters from each region of the input image. In the second stage, the operations of *integration* and *selection* are performed in parallel. The integration pathway combines information from motion energy filters tuned to different directions and spatial and temporal frequencies to compute the local evidence in favor of a particular velocity. The selection pathway weights each region of the image according to the amount of evidence for a particular velocity that region contains. In the third stage, the global evidence for a visual target moving at a particular velocity $v_k(t)$ is computed as a sum over the product of the outputs of the integration and selection pathways:

$$v_k(t) = \sum_{x,y} I_k(x,y,t) S_k(x,y,t) \tag{1}$$

where $I_k(x,y,t)$ is the local evidence for velocity $k$ computed by the integration pathway from region $(x,y)$ at time $t$, and $S_k(x,y,t)$ is the weight assigned by the selection pathway to that region.

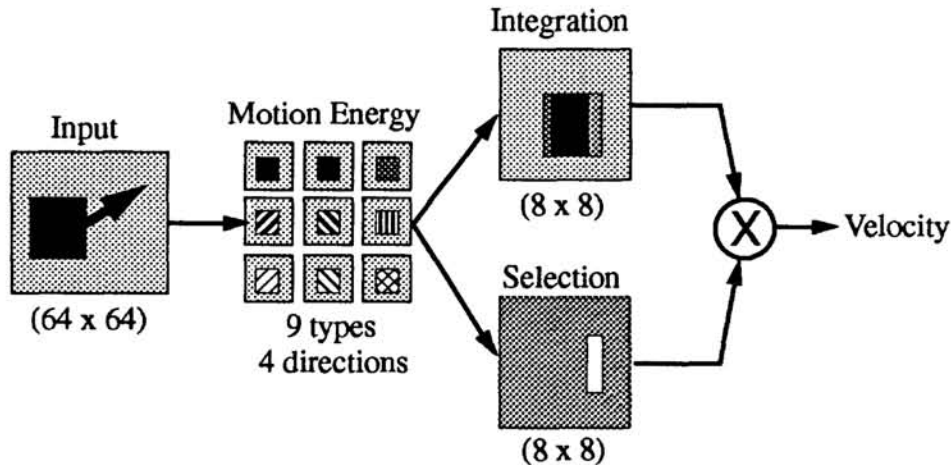

Figure 1: Diagram of motion processing model. Processing proceeds from left to right in the model, but the integration and selection stages operate in parallel. Shading within the boxes indicates different levels of activity at each stage. The responses shown in the diagram are intended to be *indicative* of the responses at different stages of the model but do not represent actual responses from the model.

## 2.1   LOCAL MOTION ESTIMATES

The first stage of processing is based on the motion energy model (Adelson and Bergen 1985, Watson 1985). This model relies on the observation that an intensity edge moving at a constant velocity produces a line at a particular orientation in space-time. This means that an oriented space-time filter will respond most strongly to objects moving at a particular velocity.[1] A motion energy filter uses the squared outputs of a quadrature pair (90° out of phase) of oriented filters to produce a phase independent local velocity estimate. The motion energy model was selected as a biologically plausible model of motion processing in mammalian V1, based primarily on the similarity of responses of simple and complex cells in cat area V1 to the output of different stages of the motion energy model (Heeger 1992, Grywacz and Yuille 1990, Emerson 1987).

The particular filters used in our model had spatial responses similar to a two-dimensional Gabor filter, with the physiologically more plausible temporal responses suggested by Adelson and Bergen (1985). The motion energy layer was divided into a grid of 49 by 49 receptive field locations and at each grid location there were filters tuned to four different directions of motion (up, down, left, and right). For each direction of motion there were nine different filters representing combinations of three spatial and three temporal frequencies. The filter center frequency spacings were 1 octave spatially and 1.5 octaves temporally. The filter parameters and spacings were chosen to be physiologically realistic, and were *fixed* during training of the model. In addition, there was a correspondence between the size of the filter

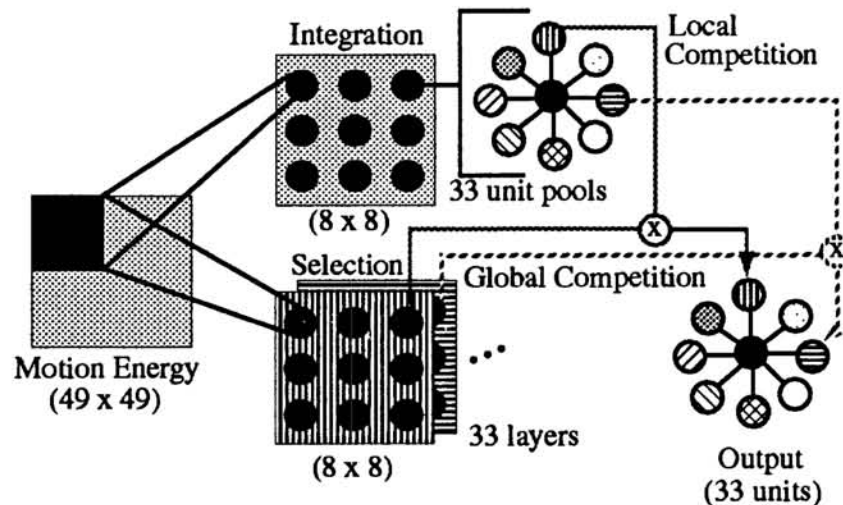

Figure 2: Diagram of integration and selection processing stages. Different shadings for units in the integration and output pools correspond to different directions of motion. Only two of the selection layers are shown and the backgrounds of these layers are shaded to match their corresponding integration and output units. See text for description of architecture.

receptive fields and the spatial frequency tuning of the filters with lower frequency filters having larger spatial extent to their receptive fields. This is also similar to what has been found in visual cortex (Maunsell and Newsome, 1987).

The input intensity image is first filtered with a difference of gaussians filter which is a simplification of retinal processing and provides smoothing and contrast enhancement. Each motion energy filter is then convolved with the smoothed input image producing 36 motion energy responses at each location in the receptive field grid which serve as the input to the next stage of processing.

## 2.2   INTEGRATION AND SELECTION

The integration and selection pathways are both implemented as locally connected networks with a single layer of weights. The integration pathway can be thought of as a layer of units organized into a grid of 8 by 8 receptive field locations (figure 2). Units at each receptive field location look at all 36 motion energy measurements from each location within a 9 by 9 region of the motion energy receptive field grid. Adjacent receptive field locations receive input from overlapping regions of the motion energy layer.

At each receptive field location in the integration layer there is a pool of 33 integration units (9 units in one of these pools are shown in figure 2). These units represent motion in 8 different directions with units representing four different speeds for each direction plus a central unit indicating no motion. These units form a log polar representation of the local velocity at that receptive field location, since as one moves out along any "arm" of the pool of units each unit represents a speed twice as large as the preceding unit in that arm. All of the integration pools share a common set

of weights, so in the final trained model all compute the same function.

The activity of an integration unit (which lies between 0 and 1) represents the amount of local support for the corresponding velocity. *Local competition* between the units in each integration pool enforces the important constraint that *each integration pool can only provide strong support for one velocity*. The competition is enforced using a *softmax* non-linearity: If $I'_k(x, y, t)$ represents the net input to unit $k$ in one of the integration pools, the state of that unit is computed as

$$I_k(x, y, t) = e^{I'_k(x,y,t)} / \sum_j e^{I'_j(x,y,t)}.$$

Note that the summation is performed over all units within a single pool, all of which share the same $(x, y)$ receptive field location.

The output of the model is also represented by a pool of 33 units, organized in the same way as each pool of integration units. The state of each unit in the output pool represents the global evidence within the entire image supporting a particular velocity. The state of each of these output units $v_k(t)$ is computed as the weighted sum of the state of the corresponding integration unit in all 64 integration receptive field locations (equation (1)). The weights assigned to each receptive field location are computed by the state of the corresponding selection unit (figure 2). Although the activity of output units can be treated as evidence for a particular velocity, the activity across the entire pool of units forms a distributed representation of a continuous range of velocities (*i.e.* activity split between two adjacent units represents a velocity between the optimal velocities of those two units).

The selection units are also organized into a grid of 8 by 8 receptive field locations which are in one to one correspondence with the integration receptive field locations (figure 2). However, it is convenient to think of the selection units as being organized not as a single layer of units but rather as 33 layers of units, one for each output unit. In each layer of selection units, there is one unit for each receptive field location. Two of the selection layers are shown in figure 2. The layer with the vertically shaded background corresponds to the output unit for upward motion (also shaded with vertical stripes) and states of units in this selection layer weight the states of upward motion units in each integration pool (again shaded vertically).

There is *global competition* among all of the units in each selection layer. Again this is implemented using a softmax non-linearity: If $S'_k(x, y, t)$ is the net input to a selection unit in layer $k$, the state of that unit is computed as

$$S_k(x, y, t) = e^{S'_k(x,y,t)} / \sum_{x',y'} e^{S'_k(x',y',t)}.$$

Note that unlike the integration case, the summation in this case is performed over all receptive field locations. This global competition enforces the second important constraint in the model, that *the total amount of support for each velocity across the entire image cannot exceed one*. This constraint, combined with the fact that the integration unit outputs can never exceed 1 ensures that the states of the output units are constrained to be between 0 and 1 and can be interpreted as the global support within the image for each velocity, as stated earlier.

The combination of global competition in the selection layers and local competition within the integration pools means that the only way to produce strong support for

a particular output velocity is for the corresponding selection network to focus all its support on regions that strongly support that velocity. This allows the selection network to learn to estimate how useful information in different regions of an image is for predicting velocities within the visual scene. The weights of both the selection and integration networks are adapted in parallel as is discussed next.

## 2.3   OBJECTIVE FUNCTION AND TRAINING

The outputs of the integration and selection networks in the final trained model are combined as in equation (1), so that the final outputs represent the global support for each velocity within the image. During training of the system however, the outputs of each pool of integration units are treated as if each were an independent estimate of support for a particular velocity. If a training image sequence contains an object moving at velocity $v_k$ then the target for the corresponding output unit is set to 1, otherwise it is set to 0. The system is then trained to maximize the likelihood of generating the targets:

$$\log L = \sum_t \sum_k \log \left( \sum_{x,y} S_k(x,y,t) \exp\left[-(v_k - I_k(x,y,t))^2\right] \right) \qquad (2)$$

To optimize this objective, each integration output $I_k(x,y,t)$ is compared to the target $v_k$ directly, and the outputs closest to the target value are assigned the most *responsibility* for that target, and hence receive the largest error signal. At the same time, the selection network states are trained to try and estimate from the input alone (*i.e.* the local motion measurements), which integration outputs are most accurate. This interpretation of the system during training is identical to the interpretation given to the *mixture of experts* (Nowlan, 1990) and the same training procedure was used. Each pool of integration units functions like an expert network, and each layer of selection units functions like a gating network.

There are, however, two important differences between the current system and the mixture of experts. First, this system uses multiple gating networks rather than a single one, allowing the system to represent more than a single velocity within an image. Second, in the mixture of experts, each expert network has an independent set of weights and essentially learns to compute a different function (usually different functions of the same input). In the current model, each pool of integration units shares the same set of weights and is constrained to compute the same function. The effect of the training procedure in this system is to bias the computations of the integration pools to favor certain types of local image features (for example, the integration stage may only make reliable velocity estimates in regions of shear or discontinuities in velocity). The selection networks learn to identify which features the integration stage is looking for, and to weight image regions most heavily which contain these kinds of features.

## 3   RESULTS AND DISCUSSION

The system was trained using 500 image sequences containing 64 frames each. These training image sequences were generated by randomly selecting one or two visual

targets for each sequence and moving these targets through randomly selected trajectories. The targets were rectangular patches that varied in size, texture, and intensity. The motion trajectories all began with the objects stationary and then one or both objects rapidly accelerated to constant velocities maintained for the remainder of the trajectory. Targets moved in one of 8 possible directions, at speeds ranging between 0 and 2.5 pixels per unit of time. In training sequences containing multiple targets, the targets were permitted to overlap (targets were assigned to different depth planes at random) and the upper target was treated as opaque in some cases and partially transparent in other cases. The system was trained using a conjugate gradient descent procedure until the response of the system on the training sequences deviated by less than 1% on average from the desired response.

The performance of the trained system was tested using a separate set of 50 test image sequences. These sequences contained 10 novel visual targets with random trajectories generated in the same manner as the training sequences. The responses on this test set remained within 2.5% of the desired response, with the largest errors occurring at the highest velocities. Several of these test sequences were designed so that targets contained edges oriented obliquely to the direction of motion, demonstrating the ability of the model to deal with aspects of the aperture problem. In addition, only small, transient increases in error were observed when two moving objects intersected, whether these objects were opaque or partially transparent.

A more challenging test of the system was provided by presenting the system with "plaid patterns" consisting of two square wave gratings drifting in different directions (Adelson and Movshon, 1982). Human observers will sometimes see a single coherent motion corresponding to the intersection of constraints (IOC) direction of the two grating motions, and sometimes see the two grating motions separately, as one grating sliding through the other. The percept reported can be altered by changing the contrast of the regions where the two gratings intersect relative to the contrast of the grating itself (Stoner *et al*, 1990). We found that for most grating patterns the model reliably reported a single motion in the IOC direction, but by manipulating the intensity of the intersection regions it was possible to find regions where the model would report the motion of the two gratings separately. Coherent grating motion was reported when the model tended to select most strongly image regions corresponding to the intersections of the gratings, while two motions were reported when the regions between the grating intersections were strongly selected.

We also explored the response properties of selection and integration units in the trained model using drifting sinusoidal gratings. These stimuli were chosen because they have been used extensively in exploring the physiological response properties of visual motion neurons in cortical visual areas (Albright 1992, Maunsell and Newsome 1987). Integration units tended to be tuned to a fairly narrow band of velocities over a broad range of spatial frequencies, like many MT cells (Maunsell and Newsome, 1987). The selection units had quite different response properties. They responded primarily to velocity shear (neighboring regions of differing velocity) and to flicker (temporal frequency) rather than true velocity. Cells with many of these properties are also common in MT (Maunsell and Newsome, 1987). A final important difference between the integration and selection units is their response to whole field motion. Integration units tend to have responses which are somewhat enhanced by whole field motion in their preferred direction, while selection unit

responses are generally suppressed by whole field motion. This difference is similar to the recent observation that area MT contains two classes of cell, one whose responses are suppressed by whole field motion, while responses of the second class are not suppressed (Born and Tootell, 1992).

Finally, the model that we have proposed is built on the premise of an active mechanism for selecting subsets of unit responses to integrate over. While this is a common aspect of many accounts of attentional phenomena, we suggest that active selection may represent a fundamental aspect of cortical processing that occurs with many pre-attentive phenomena, such as motion processing.

## Footnotes

*Current address, Synaptics Inc., 2698 Orchard Parkway, San Jose, CA 95134.

[1]These filters actually respond most strongly to a narrow band of spatial frequencies (SF) and temporal frequencies (TF), which represent a range of velocities, $v = TF/SF$.

## References

Adelson, E. H. and Bergen, J. R. (1985) Spatiotemporal energy models for the perception of motion. *J. Opt. Soc. Am.* A, **2**, 284-299.

Adelson, M. and Movshon, J. A. (1982) Phenomenal coherence of moving visual patterns. *Nature*, **300**, 523-525.

Albright, T. D. (1992) Form-cue invariant motion processing in primate visual cortex. *Science.* **255**, 1141-1143.

Born, R. T. and Tootell, R. B. H. (1992) Segregation of global and local motion processing in primate middle temporal visual area. *Nature*, **357**, 497-500.

Emerson, R.C., Citron, M.C., Vaughn W.J., Klein, S.A. (1987) Nonlinear directionally selective subunits in complex cells of cat striate cortex. *J. Neurophys.* **58**, 33-65.

Grzywacz, N.M. and Yuille, A.L. (1990) A model for the estimate of local image velocity by cells in the visual cortex. *Proc. R. Soc. Lond.* B **239**, 129-161.

Heeger, D.J. (1987) Model for the extraction of image flow. *J. Opt. Soc. Am.* A **4**, 1455-1471.

Heeger, D.J. (1992) Normalization of cell responses in cat striate cortex. *Visual Neuroscience*, in press.

Horn, B.K.P. and Schunk, B.G. (1981) Determining optical flow. *Artificial Intelligence* **17**, 185-203.

Maunsell J.H.R. and Newsome, W.T. (1987) Visual processing in monkey extrastriate cortex. *Ann. Rev. Neurosci.* **10**, 363-401.

Nowlan, S.J. (1990) Competing experts: An experimental investigation of associative mixture models. Technical Report CRG-TR-90-5, Department of Computer Science, University of Toronto.

Nagel, H.H. (1987) On the estimation of optical flow: relations between different approaches and some new results. *Artificial Intelligence* **33**, 299-324.

Stoner G.R., Albright T.D., Ramachandran V.S. (1990) Transparency and coherence in human motion perception. *Nature* **344**, 153-155.

Watson, A.B. and Ahumada, A.J. (1985) Model of human visual-motion sensing. *J. Opt. Soc. Am.* A, **2**, 322-342.